# Using Machine Learning to Break Visual Human Interaction Proofs (HIPs)

**Kumar Chellapilla**
Microsoft Research
One Microsoft Way
Redmond, WA 98052
*kumarc@microsoft.com*

**Patrice Y. Simard**
Microsoft Research
One Microsoft Way
Redmond, WA 98052
*patrice@microsoft.com*

## Abstract

Machine learning is often used to automatically solve human tasks. In this paper, we look for tasks where machine learning algorithms are not as good as humans with the hope of gaining insight into their current limitations. We studied various Human Interactive Proofs (HIPs) on the market, because they are systems designed to tell computers and humans apart by posing challenges presumably too hard for computers. We found that most HIPs are pure recognition tasks which can easily be broken using machine learning. The harder HIPs use a combination of segmentation and recognition tasks. From this observation, we found that building segmentation tasks is the most effective way to confuse machine learning algorithms. This has enabled us to build effective HIPs (which we deployed in MSN Passport), as well as design challenging segmentation tasks for machine learning algorithms.

## 1 Introduction

The OCR problem for high resolution printed text has virtually been solved 10 years ago [1]. On the other hand, cursive handwriting recognition today is still too poor for most people to rely on. Is there a fundamental difference between these two seemingly similar problems?

To shed more light on this question, we study problems that have been designed to be difficult for computers. The hope is that we will get some insight on what the stumbling blocks are for machine learning and devise appropriate tests to further understand their similarities and differences.

Work on distinguishing computers from humans traces back to the original Turing Test [2] which asks that a human distinguish between another human and a machine by asking questions of both. Recent interest has turned to developing systems that allow a computer to distinguish between another computer and a human. These systems enable the construction of automatic filters that can be used to prevent automated scripts from utilizing services intended for humans [4]. Such systems have been termed Human Interactive Proofs (HIPs) [3] or Completely Automated Public Turing Tests to Tell Computers and Humans Apart (CAPTCHAs) [4]. An overview of the work in this area can be found in [5]. Construction of HIPs that are of practical value is difficult because it is not sufficient to develop challenges at

which humans are somewhat more successful than machines. This is because the cost of failure for an automatic attacker is minimal compared to the cost of failure for humans. Ideally a HIP should be solved by humans more than 80% of the time, while an automatic script with reasonable resource use should succeed less than 0.01% of the time. This latter ratio (1 in 10,000) is a function of the cost of an automatic trial divided by the cost of having a human perform the attack.

This constraint of generating tasks that are failed 99.99% of the time by all automated algorithms has generated various solutions which can easily be sampled on the internet. Seven different HIPs, namely, Mailblocks, MSN (before April 28th, 2004), Ticketmaster, Yahoo, Yahoo v2 (after Sept'04), Register, and Google, will be given as examples in the next section. We will show in Section 3 that machine-learning-based attacks are far more successful than 1 in 10,000. Yet, some of these HIPs are harder than others and could be made even harder by identifying the recognition and segmentation parts, and emphasizing the latter. Section 4 presents examples of more difficult HIPs which are much more respectable challenges for machine learning, and yet surprisingly easy for humans. The final section discusses a (known) weakness of machine learning algorithms and suggests designing simple artificial datasets for studying this weakness.

## 2  Examples of HIPs

The HIPs explored in this paper are made of characters (or symbols) rendered to an image and presented to the user. Solving the HIP requires identifying all characters in the correct order. The following HIPs can be sampled from the web:

**Mailblocks:** While signing up for free email service with mailblocks (www.mailblocks.com), you will find HIP challenges of the type:

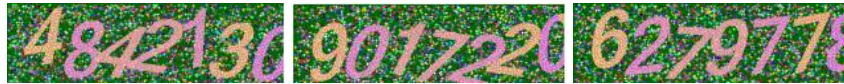

**MSN:** While signing up for free e-mail with MSN Hotmail (www.hotmail.com), you will find HIP challenges of the type:

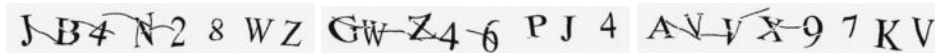

**Register.com:** While requesting a whois lookup for a domain at www.register.com, you will HIP challenges of the type:

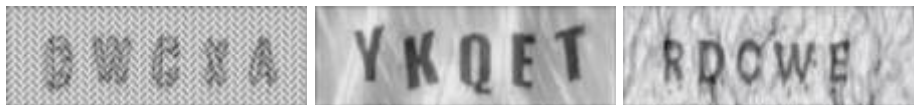

**Yahoo!/EZ-Gimpy (CMU):** While signing up for free e-mail service with Yahoo! (www.yahoo.com), you will receive HIP challenges of the type:

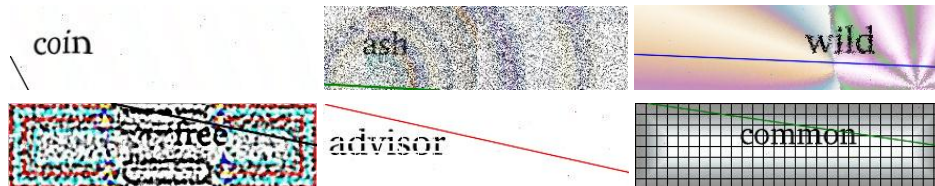

**Yahoo! (version 2):** Starting in August 2004, Yahoo! introduced their second generation HIP. Three examples are presented below:

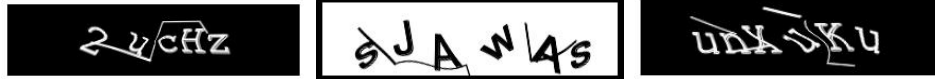

**Ticketmaster:** While looking for concert tickets at **www.ticketmaster.com**, you will receive HIP challenges of the type:

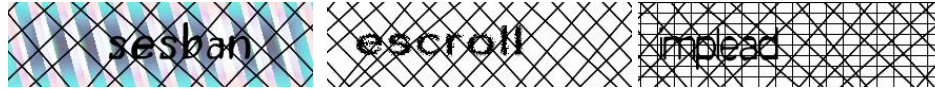

**Google/Gmail:** While signing up for free e-mail with Gmail at www.google.com, one will receive HIP challenges of the type:

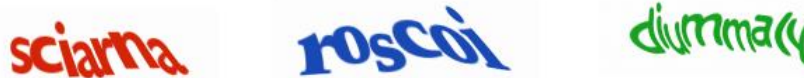

While solutions to Yahoo HIPs are common English words, those for ticketmaster and Google do not necessarily belong to the English dictionary. They appear to have been created using a phonetic generator [8].

## 3 Using machine learning to break HIPs

Breaking HIPs is not new. Mori and Malik [7] have successfully broken the EZ-Gimpy (92% success) and Gimpy (33% success) HIPs from CMU. Our approach aims at an automatic process for solving multiple HIPs with minimum human intervention, using machine learning. In this paper, our main goal is to learn more about the common strengths and weaknesses of these HIPs rather than to prove that we can break any one HIP in particular with the highest possible success rate. We have results for six different HIPs: EZ-Gimpy/Yahoo, Yahoo v2, mailblocks, register, ticketmaster, and Google.

To simplify our study, we will not be using language models in our attempt to break HIPs. For example, there are only about 600 words in the EZ-Gimpy dictionary [7], which means that a random guess attack would get a success rate of 1 in 600 (more than enough to break the HIP, i.e., greater than 0.01% success). HIPs become harder when no language model is used. Similarly, when a HIP uses a language model to generate challenges, success rate of attacks can be significantly improved by incorporating the language model. Further, since the language model is not common to all HIPs studied, it was not used in this paper.

Our generic method for breaking all of these HIPs is to write a custom algorithm to locate the characters, and then use machine learning for recognition. Surprisingly, segmentation, or finding the characters, is simple for many HIPs which makes the process of breaking the HIP particularly easy. Gimpy uses a single constant predictable color (black) for letters even though the background color changes. We quickly realized that once the segmentation problem is solved, solving the HIP becomes a pure recognition problem, and it can trivially be solved using machine learning. Our recognition engine is based on neural networks [6][9]. It yielded a 0.4% error rate on the MNIST database, uses little memory, and is very fast for recognition (important for breaking HIPs).

For each HIP, we have a segmentation step, followed by a recognition step. It should be stressed that we are not trying to solve every HIP of a given type i.e., our goal is not 100% success rate, but something efficient that can achieve much better than 0.01%.

In each of the following experiments, 2500 HIPs were hand labeled and used as follows (a) recognition (1600 for training, 200 for validation, and 200 for testing), and (b) segmentation (500 for testing segmentation). For each of the five HIPs, a convolution neural network, identical to the one described in [6], was trained and tested on gray level character images centered on the guessed character positions (see below). The trained neural network became the recognizer.

## 3.1 Mailblocks

To solve the HIP, we select the red channel, binarize and erode it, extract the largest connected components (CCs), and breakup CCs that are too large into two or three adjacent CCs. Further, vertically overlapping half character size CCs are merged. The resulting rough segmentation works most of the time. Here is an example:

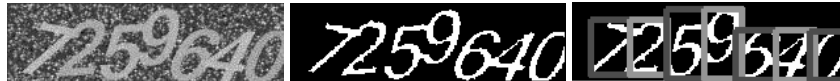

For instance, in the example above, the NN would be trained, and tested on the following images:

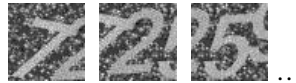

The end-to-end success rate is 88.8% for segmentation, 95.9% for recognition (given correct segmentation), and $(0.888)*(0.959)^7 = 66.2\%$ total. Note that most of the errors come from segmentation, even though this is where all the custom programming was invested.

## 3.2 Register

The procedure to solve HIPs is very similar. The image was smoothed, binarized, and the largest 5 connected components were identified. Two examples are presented below:

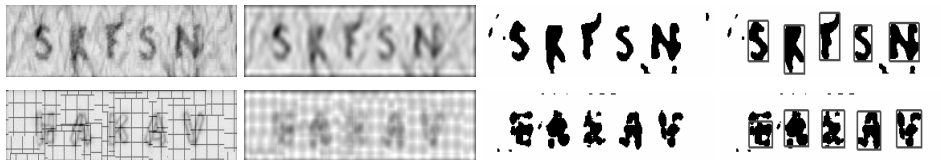

The end-to-end success rate is 95.4% for segmentation, 87.1% for recognition (given correct segmentation), and $(0.954)*(0.871)^5 = 47.8\%$ total.

## 3.3 Yahoo/EZ-Gimpy

Unlike the mailblocks and register HIPs, the Yahoo/EZ-Gimpy HIPs are richer in that a variety of backgrounds and clutter are possible. Though some amount of text warping is present, the text color, size, and font have low variability. Three simple segmentation algorithms were designed with associated rules to identify which algorithm to use. The goal was to keep these simple yet effective:

a) **No mesh**: Convert to grayscale image, threshold to black and white, select large CCs with sizes close to HIP char sizes. One example:

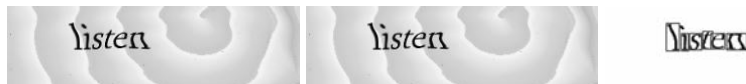

b) **Black mesh**: Convert to grayscale image, threshold to black and white, remove vertical and horizontal line pixels that don't have neighboring pixels, select large CCs with sizes close to HIP char sizes. One example:

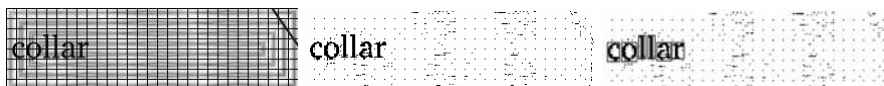

c) **White mesh**: Convert to grayscale image, threshold to black and white, add black pixels (in white line locations) if there exist neighboring pixels, select large CCs with sizes close to HIP char sizes. One example:

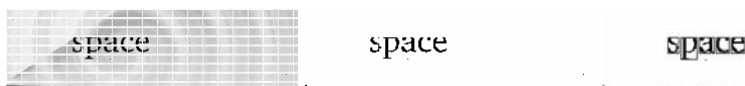

Tests for black and white meshes were performed to determine which segmentation algorithm to use. The end-to-end success rate was 56.2% for segmentation (38.2% came from a), 11.8% from b), and 6.2% from c), 90.3% for recognition (given correct segmentation), and $(0.562)*(0.903)^{4.8} = 34.4\%$ total. The average length of a Yahoo HIP solution is 4.8 characters.

## 3.4 Ticketmaster

The procedure that solved the Yahoo HIP is fairly successful at solving some of the ticket master HIPs. These HIPs are characterized by cris-crossing lines at random angles clustered around 0, 45, 90, and 135 degrees. A multipronged attack as in the Yahoo case (section 3.3) has potential. In the interests of simplicity, a single attack was developed: Convert to grayscale, threshold to black and white, up-sample image, dilate first then erode, select large CCs with sizes close to HIP char sizes. One example:

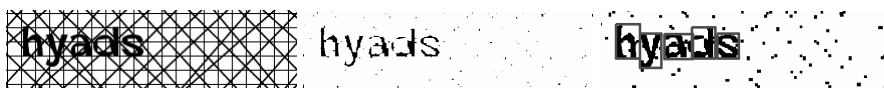

The dilate-erode combination causes the lines to be removed (along with any thin objects) but retains solid thick characters. This single attack is successful in achieving an end-to-end success rate of 16.6% for segmentation, the recognition rate was 82.3% (in spite of interfering lines), and $(0.166)*(0.823)^{6.23} = 4.9\%$ total. The average HIP solution length is 6.23 characters.

## 3.5 Yahoo version 2

The second generation HIP from Yahoo had several changes: a) it did not use words from a dictionary or even use a phonetic generator, b) it uses only black and white colors, c) uses both letters and digits, and d) uses connected lines and arcs as clutter. The HIP is somewhat similar to the MSN/Passport HIP which does not use a dictionary, uses two colors, uses letters and digits, and background and foreground arcs as clutter. Unlike the MSN/Passport HIP, several different fonts are used. A single segmentation attack was developed: Remove 6 pixel border, up-sample, dilate first then erode, select large CCs with sizes close to HIP char sizes. The attack is practically identical to that used for the ticketmaster HIP with different preprocessing stages and slightly modified parameters. Two examples:

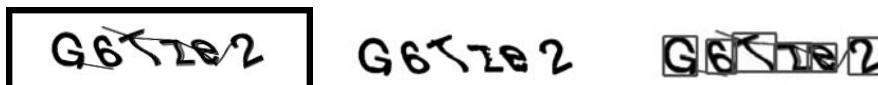

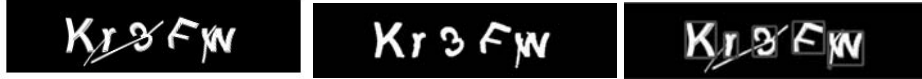

This single attack is successful in achieving an end-to-end success rate of 58.4% for segmentation, the recognition rate was 95.2%, and $(0.584)*(0.952)^5 = 45.7\%$ total. The average HIP solution length is 5 characters.

### 3.6 Google/GMail

The Google HIP is unique in that it uses only image warp as a means of distorting the characters. Similar to the MSN/Passport and Yahoo version 2 HIPs, it is also two color. The HIP characters are arranged closed to one another (they often touch) and follow a curved baseline. The following very simple attack was used to segment Google HIPs: Convert to grayscale, up-sample, threshold and separate connected components.

a) 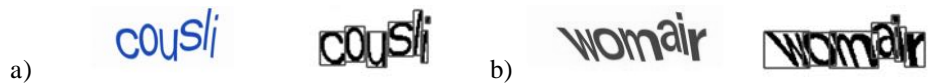 b)

This very simple attack gives an end-to-end success rate of 10.2% for segmentation, the recognition rate was 89.3%, giving $(0.102)*(0.893)^{6.5} = 4.89\%$ total probability of breaking a HIP. Average Google HIP solution length is 6.5 characters. This can be significantly improved upon by judicious use of dilate-erode attack. A direct application doesn't do as well as it did on the ticketmaster and yahoo HIPs (because of the shear and warp of the baseline of the word). More successful and complicated attacks might estimate and counter the shear and warp of the baseline to achieve better success rates.

## 4   Lessons learned from breaking HIPs

From the previous section, it is clear that most of the errors come from incorrect segmentations, even though most of the development time is spent devising custom segmentation schemes. This observation raises the following questions: Why is segmentation a hard problem? Can we devise harder HIPs and datasets? Can we build an automatic segmentor? Can we compare classification algorithms based on how useful they are for segmentation?

### 4.1   The segmentation problem

As a review, segmentation is difficult for the following reasons:
1. Segmentation is computationally expensive. In order to find valid patterns, a recognizer must attempt recognition at many different candidate locations.
2. The segmentation function is complex. To segment successfully, the system must learn to identify which patterns are valid among the set of all possible valid and non-valid patterns. This task is intrinsically more difficult than classification because the space of input is considerably larger. Unlike the space of valid patterns, the space of non-valid patterns is typically too vast to sample. This is a problem for many learning algorithms which yield too many false positives when presented non-valid patterns.
3. Identifying valid characters among a set of valid and invalid candidates is a combinatorial problem. For example, correctly identifying which 8 characters among 20 candidates (assuming 12 false positives), has a 1 in 125,970 (20 choose 8) chances of success by random guessing.

## 4.2 Building better/harder HIPs

We can use what we have learned to build better HIPs. For instance the HIP below was designed to make segmentation difficult and a similar version has been deployed by MSN Passport for hotmail registrations (www.hotmail.com):

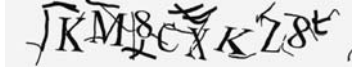

The idea is that the additional arcs are themselves good candidates for false characters. The previous segmentation attacks would fail on this HIP. Furthermore, simple change of fonts, distortions, or arc types would require extensive work for the attacker to adjust to. We believe HIPs that emphasize the segmentation problem, such as the above example, are much stronger than the HIPs we examined in this paper, which rely on recognition being difficult. Pushing this to the extreme, we can easily generate the following HIPs:

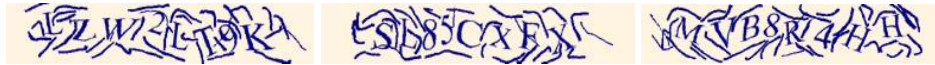

Despite the apparent difficulty of these HIPs, humans are surprisingly good at solving these, indicating that humans are far better than computers at segmentation. This approach of adding several competing false positives can in principle be used to automatically create difficult segmentation problems or benchmarks to test classification algorithms.

## 4.3 Building an automatic segmentor

To build an automatic segmentor, we could use the following procedure. Label characters based on their correct position and train a recognizer. Apply the trained recognizer at all locations in the HIP image. Collect all candidate characters identified with high confidence by the recognizer. Compute the probability of each combination of candidates (going from left to right), and output the solution string with the highest probability. This is better illustrated with an example.

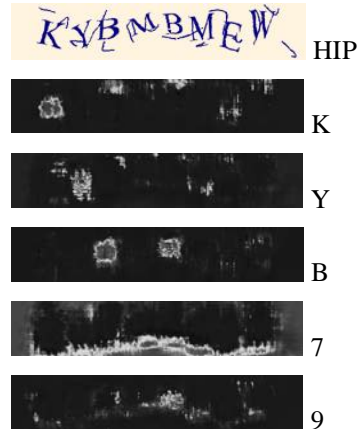

Consider the following HIP (to the right). The trained neural network has these maps (warm colors indicate recognition) that show that K, Y, and so on are correctly identified. However, the maps for 7 and 9 show several false positives. In general, we would get the following color coded map for all the different candidates:

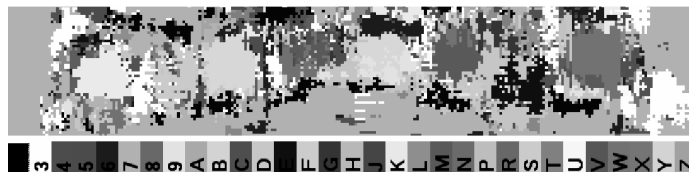

With a threshold of 0.5 on the network's outputs, the map obtained is:

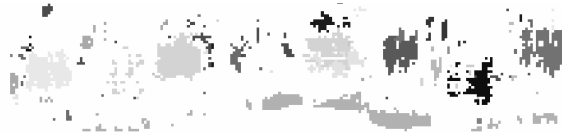

We note that there are several false positives for each true positive. The number of false positives per true positive character was found to be between 1 and 4, giving a 1 in C(16,8) = 12,870 to 1 in C(32,8) = 10,518,300 random chance of guessing the correct segmentation for the HIP characters. These numbers can be improved upon by constraining solution strings to flow sequentially from left to right and by restricting overlap. For each combination, we compute a probability by multiplying the 8 probabilities of the classifier for each position. The combination with the highest probability is the one proposed by the classifier. We do not have results for such an automatic segmentor at this time. It is interesting to note that with such a method a classifier that is robust to false positives would do far better than one that is not. This suggests another axis for comparing classifiers.

## 5   Conclusion

In this paper, we have successfully applied machine learning to the problem of solving HIPs. We have learned that decomposing the HIP problem into segmentation and recognition greatly simplifies analysis. Recognition on even unprocessed images (given segmentation is a solved) can be done automatically using neural networks. Segmentation, on the other hand, is the difficulty differentiator between weaker and stronger HIPs and requires custom intervention for each HIP. We have used this observation to design new HIPs and new tests for machine learning algorithms with the hope of improving them.

### Acknowledgements

We would like to acknowledge Chau Luu and Eric Meltzer for their help with labeling and segmenting various HIPs. We would also like to acknowledge Josh Benaloh and Cem Paya for stimulating discussions on HIP security.

### References

[1] Baird HS (1992), "Anatomy of a versatile page reader," *IEEE Pro.*, v.80, pp. 1059-1065.

[2] Turing AM (1950), "Computing Machinery and Intelligence," *Mind*, 59:236, pp. 433-460.

[3] *First Workshop on Human Interactive Proofs*, Palo Alto, CA, January 2002.

[4] Von Ahn L, Blum M, and Langford J, *The Captcha Project.* http://www.captcha.net

[5] Baird HS and Popat K (2002) "Human Interactive Proofs and Document Image Analysis," *Proc. IAPR 2002 Workshop on Document Analysis Systerms*, Princeton, NJ.

[6] Simard PY, Steinkraus D, and Platt J, (2003) "Best Practice for Convolutional Neural Networks Applied to Visual Document Analysis," in *International Conference on Document Analysis and Recognition* (ICDAR), pp. 958-962, IEEE Computer Society, Los Alamitos.

[7] Mori G, Malik J (2003), "Recognizing Objects in Adversarial Clutter: Breaking a Visual CAPTCHA," *Proc. of the Computer Vision and Pattern Recognition (CVPR) Conference*, IEEE Computer Society, vol.1, pages:I-134 - I-141, June 18-20, 2003

[8] Chew, M. and Baird, H. S. (2003), "BaffleText: a Human Interactive Proof," *Proc., 10th IS&T/SPIE Document Recognition & Retrieval Conf.,* Santa Clara, CA, Jan. 22.

[9] LeCun Y, Bottou L, Bengio Y, and Haffner P, "Gradient-based learning applied to document recognition,' *Proceedings of the IEEE*, Nov. 1998.